# A Discriminative Latent Model of Image Region and Object Tag Correspondence

**Yang Wang**\*
Department of Computer Science
University of Illinois at Urbana-Champaign
yangwang@uiuc.edu

**Greg Mori**
School of Computing Science
Simon Fraser University
mori@cs.sfu.ca

## Abstract

We propose a discriminative latent model for annotating images with unaligned object-level textual annotations. Instead of using the bag-of-words image representation currently popular in the computer vision community, our model explicitly captures more intricate relationships underlying visual and textual information. In particular, we model the mapping that translates image regions to annotations. This mapping allows us to relate image regions to their corresponding annotation terms. We also model the overall scene label as latent information. This allows us to cluster test images. Our training data consist of images and their associated annotations. But we do not have access to the ground-truth region-to-annotation mapping or the overall scene label. We develop a novel variant of the latent SVM framework to model them as latent variables. Our experimental results demonstrate the effectiveness of the proposed model compared with other baseline methods.

## 1  Introduction

Image understanding is a central problem in computer vision that has been extensively studied in the forms of various types of tasks. Some previous work focuses on classifying an image with a single label [6]. Others go beyond single labels and assign a list of annotations to an image [1, 10, 21]. Recently, efforts have been made to combine various tasks (i.e. classification, annotation, segmentation, etc) together to achieve a more complete understanding of an image [11, 12]. In this paper, we consider the problem of image understanding with unaligned textual annotations. In particular, we focus on the scenario where the annotations represent the names of the objects present in an image. The input to our learning algorithm is a set of images with unaligned textual annotations (object names). Our goal is to learn a model to predict the annotation (i.e. object names) for a new image. As a by-product, our model also roughly localizes the image regions corresponding to the annotation, see Fig. 1. The main contribution of this paper is the development of a model that incorporates this object annotation to image region correspondence in a discriminative framework.

In the computer vision literature, there has been a lot of work on exploiting images and their associated textual information. Barnard et al. [1] predict words associated with whole images or regions by learning a joint distribution of image regions and words. Berg et al. [3] learn to name faces appearing in news pictures by learning a probabilistic model of face appearances, names, and textual contexts. Wang et al. [21] use a learned bag-of-words topic model to simultaneously classify and annotate images. Loeff et al. [13] discover scenes by exploiting the correlation between images and their annotations. Some recent work towards total scene understanding [11, 12] tries to build sophisticated generative models that jointly perform several tasks, e.g. scene classification, object recognition, image annotation, and image segmentation.

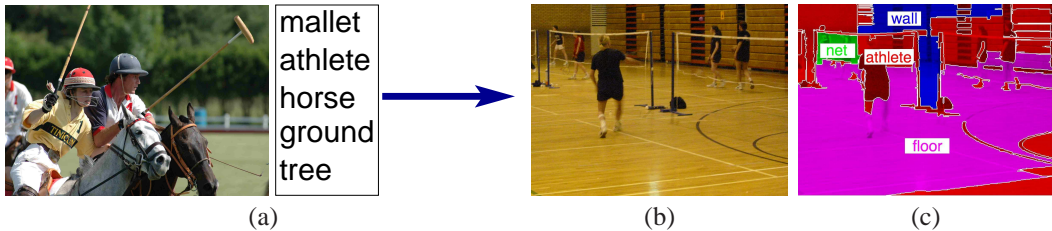

*Figure 1:* Our goal is to learn a model using images and their associated unaligned textual object annotations (a) as the training data. Given a new image (b), we can use the model to predict its textual annotations and roughly localize image regions corresponding to each of the annotation terms (c).

Most of the previous work uses fairly crude "bag-of-words" models, treating image features (extracted from either segmented regions or local interest points) and textual annotations as unordered entities and looking at their co-occurrence statistics. Very little work explicitly models more detailed relationships between image regions and annotations that are obvious to humans. For example, if an image is over-segmented into a large number of segments, each segment typically only corresponds to at most one object. However, most of the previous work ignores this constraint and allows an image region being used as evidence to explain different objects mentioned by the annotations. In this paper, we present a discriminative latent model that captures image regions, textual annotations, mappings between visual and textual information, and overall scene labels in a more explicit manner. Some work [1, 3] tries to incorporate the mapping information into a generative model. However due to the limitation of the machine learning tools used in those work, they did not properly enforce the aforementioned constraint on how image regions are mapped to annotations. There is also work [2] on augmenting training data with this mapping information, but it is unclear how it can be generalized on test data. With the recent advancement in learning with complex structured data [7, 18, 21, 25], we believe now it is the time for us to revisit this line of ideas and examine other modeling tools.

The work by Socher et al. [17] is the most relevant to ours. In that work, they learn to annotate and segment images by mapping image regions and textual words to a latent meaning space using context and adjective features. There are important distinctions between our work and [17]. First of all, the input to [17] is a set of images (a handful of which are manually labeled) of *a single sport category*, and a collection of news articles for that sport. The news articles are generic for that sport, and the images are not the news photographs directly associated with those news articles. Although they have experimented on applying their model on image collections with mixed sport categories, their method seems to work better with single sport category training. In contrast, the input to our learning problem is a set of images from several sport categories, together with their *associated textual annotations*. We treat the sport category as a latent variable (we call it the *scene label*) and implicitly infer it during learning.

## 2 Model

We propose a discriminative latent model that jointly captures the relationships between image segments, textual annotations, region-text correspondence, and overall image visual scene labels. Of course, only the image segments and textual annotations are observed on training data. All the other information (e.g. scene labels, the mapping between regions and annotations) are treated as latent variables in the model. A graphical illustration of our model is shown in Fig. 2.

The input to our learning module is a set of $\langle \mathbf{x}, \mathbf{y} \rangle$ pairs where $\mathbf{x}$ denotes an image, and $\mathbf{y}$ denotes the annotation associated with this image. We partition the image into $R$ regions using the segmentation algorithm in [8], i.e. $\mathbf{x} = [x_1, x_2, ..., x_R]$. For each image region $x_i$, we extract four types of visual features (see [14]): shape, texture, color, and location. Each of these feature types is vector quantized to obtain codewords for this feature type. Following [17], we use 20, 25, 40, 8 codewords for each of the four feature types, respectively. In the end, each region $x_i$ is represented as a 4-dimensional vector $x_i = (x_{i1}, x_{i2}, x_{i3}, x_{i4})$, where each $x_{ic}$ is the corresponding codeword of the $c$-th feature type for this region.

The annotation $\mathbf{y}$ of an image is represented as a binary vector $\mathbf{y} = (y_1, y_2, ..., y_V)$, where $V$ is the total number of possible annotation terms. As a terminological convention, we use "annotation" to denote the vector $\mathbf{y}$ and "annotation term" to denote each component $y_j$ of the vector. An annotation

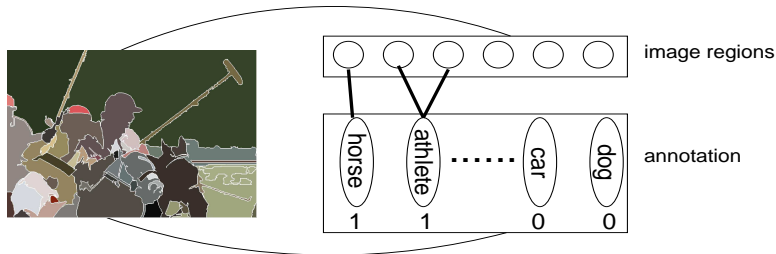

*Figure 2:* Graphical illustration of our model. An input image is segmented into several regions. The annotation of the image is represented as a 0-1 vector indicating the presence/absence of each possible annotation term. Our model captures the unobserved mapping that translate image regions to annotation terms associated with the image (e.g. horse, athlete). For annotation terms not associated with the image (e.g. car, dog), there are no mapped image regions. Our model also captures relationship between the unobserved scene label (e.g. polo) and image regions/annotations.

term $y_j$ is "active" ($y_j = 1$) if it is associated with this image, and is "inactive" ($y_j = 0$) otherwise. We further assume the number of regions of an image is larger than or equal to the number of active annotation terms for an image, i.e. $R \geq \sum_{j=1}^{V} y_j$. In this work, we assume there are no visually irrelevant annotation terms (e.g. "wind"), and there are no annotation terms (e.g. "people" and "athlete") of an image that refer to the same concept. These can be achieved by pre-processing the annotation terms with Wordnet (see [17]).

Given an image $\mathbf{x}$ and its annotation $\mathbf{y}$, we assume there is an underlying unobserved many-to-one mapping which translates $R$ image regions to each of the active annotation terms. We restrict the mapping to have the following conditions: (i) each image region is mapped to at most one annotation term. This condition will ensure that an image region is not used to explain two different annotations; (ii) an active annotation term has one or more image regions mapped to it. This condition will make sure that if an annotation term (say "building") is assigned to an image, there is at least one image region supporting this annotation term; (iii) an inactive annotation term has no image regions mapped to it. This condition will guarantee there are no image regions supporting an inactive annotation term.

More formally, we introduce a matrix $\mathbf{z} = \{z_{ij} : 1 \leq i \leq R, 1 \leq j \leq V\}$ defined in the following to represent this mapping for an image with $R$ regions:

$$z_{ij} = \begin{cases} 1 & \text{if the } i\text{-th image region is mapped to the } j\text{-th annotation term} \\ 0 & \text{otherwise} \end{cases} \tag{1}$$

We use $\mathcal{Y}$ to denote the domain of all possible assignments of $\mathbf{y}$. For a fixed annotation $\mathbf{y}$, we use $\mathcal{Z}(\mathbf{y})$ to denote the set of all possible many-to-one mappings that satisfy the conditions (i,ii,iii). It is easy to verify that any $\mathbf{z} \in \mathcal{Z}(\mathbf{y})$ can be represented using the following three sets of constraints:

$$\sum_{j} z_{ij} \leq 1, \ \forall i; \qquad \max_{i} z_{ij} = y_j, \ \forall j; \qquad z_{ij} \in \{0,1\}, \ \forall i, \ \forall j \tag{2}$$

For a given image, we also assume a discrete unobserved "scene" label $s$ which takes its value between 1 and $\mathcal{S}$. We introduce the scene label to capture the fact that the annotations of images are typically well clustered according to their underlying scenes. For example, an image of a "sailing" scene tends to have annotation terms like "athlete", "sailboat", "water", etc. However, it is not quite simple to define the vocabulary to label scenes [13]. In our work, we treat the scene label as a latent variable (hence we do not need its ground-truth label or even a vocabulary for defining it) and let the learning algorithm automatically figure out what constitutes a scene. As we will demonstrate in the experiment, the "scenes" learned by our model on a collection of sport images do match our intuitions, e.g. they roughly correspond to different sport categories in the data.

Inspired by the latent SVM [7, 25], we measure the compatibility between an image $\mathbf{x}$ and an annotation $\mathbf{y}$ using the following scoring function:

$$f_\theta(\mathbf{x}, \mathbf{y}) = \max_{s \in \mathcal{S}} \ \max_{\mathbf{z} \in \mathcal{Z}(\mathbf{y})} \theta^\top \cdot \Phi(\mathbf{x}, \mathbf{y}, \mathbf{z}, s) \tag{3}$$

where $\theta$ are the model parameters and $\Phi(\mathbf{x}, \mathbf{y}, \mathbf{z}, s)$ is a feature vector defined on $\mathbf{x}, \mathbf{y}, \mathbf{z}$ and $s$. The model parameters have three parts $\theta = \{\alpha, \beta, \gamma\}$, and $\theta^\top \cdot \Phi(\mathbf{x}, \mathbf{y}, \mathbf{z}, s)$ is defined as:

$$\theta^\top \cdot \Phi(\mathbf{x}, \mathbf{y}, \mathbf{z}, s) = \alpha^\top \phi(\mathbf{x}, \mathbf{z}) + \beta^\top \psi(\mathbf{x}, s) + \gamma^\top \varphi(\mathbf{y}, s) \tag{4}$$

The details of each of the terms in (4) are described in the following.

**Region-Annotation Matching Potential $\alpha^\top \phi(\mathbf{x}, \mathbf{z})$:** This potential function measures the compatibility of mapping image regions to their corresponding annotation terms. Recall an image region $x_i$ consists of codewords from four different feature types $x_i = (x_{i1}, x_{i2}, x_{i3}, x_{i4})$. Let $N_c$ ($c = 1, 2, 3, 4$) denotes the number of codewords of feature type $c$. The parameters $\alpha$ consist of four components $\alpha = \{\alpha^c\}_{c=1}^4$ corresponding to each of the four feature types. Each $\alpha^c$ is a matrix of size $N_c \times V$, where an entry $\alpha_{w,j}^c$ can be interpreted as the compatibility between the codeword $w$ ($1 \le w \le N_c$) of feature type $c$ and the annotation term $j$ ($1 \le j \le V$). The potential function is written as:

$$\alpha^\top \phi(\mathbf{x}, \mathbf{z}) = \sum_{c=1}^4 \sum_{i=1}^R \sum_{j=1}^V \alpha_{x_{ic},j}^c \cdot z_{ij} = \sum_{c=1}^4 \sum_{i=1}^R \sum_{w=1}^{N_c} \sum_{j=1}^V \alpha_{w,j}^c \cdot \mathbb{1}(x_{ic} = w) \cdot z_{ij} \tag{5}$$

where $\mathbb{1}(\cdot)$ is the indicator function. Note that the definition of this potential function does not involve $\mathbf{y}$ since $\mathbf{y}$ is implicitly determined by $\mathbf{z}$, i.e. $y_j = \max_i z_{ij}$.

**Image-Scene Potential $\beta^\top \psi(\mathbf{x}, s)$:** This potential function measures the compatibility between an image $\mathbf{x}$ and a scene label $s$. Similarly, the parameters $\beta$ consist of four parts $\beta = \{\beta^c\}_{c=1}^4$ corresponding to the four feature types, where an entry $\beta_{w,s}^c$ is the compatibility between the codeword $w$ of type $c$ and the scene label $s$. This potential function is written as:

$$\beta^\top \psi(\mathbf{x}, s) = \sum_{c=1}^4 \sum_{i=1}^R \beta_{x_{ic},s}^c = \sum_{c=1}^4 \sum_{i=1}^R \sum_{w=1}^{N_c} \sum_{t=1}^S \beta_{w,t}^c \cdot \mathbb{1}(x_{ic} = w) \cdot \mathbb{1}(s = t) \tag{6}$$

**Annotation-Scene Potential $\gamma^\top \varphi(\mathbf{y}, s)$:** This potential function measures the compatibility between an annotation $\mathbf{y}$ and a scene label $s$. The parameters $\gamma$ consist of $S$ components $\gamma = \{\gamma^t\}_{t=1}^S$ corresponding to each of the scene label. Each component $\gamma^t$ is a $V \times 2$ matrix, where $\gamma_{j,1}^t$ is the compatibility of setting $y_j = 1$ for the scene label $t$, and $\gamma_{j,0}^t$ is the compatibility of setting $y_j = 0$ for the scene label $t$. This potential function is written as:

$$\gamma^\top \varphi(\mathbf{y}, s) = \sum_{j=1}^V \gamma_{j,y_j}^s = \sum_{j=1}^V \sum_{t=1}^S \left( \gamma_{j,0}^t \cdot \mathbb{1}(y_j = 0) \cdot \mathbb{1}(s = t) + \gamma_{j,1}^t \cdot \mathbb{1}(y_j = 1) \cdot \mathbb{1}(s = t) \right) \tag{7a}$$

$$= \sum_{j=1}^V \sum_{t=1}^S \left( \gamma_{j,0}^t \cdot (1 - y_j) \cdot \mathbb{1}(s = t) + \gamma_{j,1}^t \cdot y_j \cdot \mathbb{1}(s = t) \right) \tag{7b}$$

The equivalence of (7a) and (7b) is due to $\mathbb{1}(y_j = 0) \equiv 1 - y_j$ and $\mathbb{1}(y_j = 1) \equiv y_j$ for $y_j \in \{0, 1\}$, which are easy to verify.

## 3 Inference

Given the model parameters $\theta = \{\alpha, \beta, \gamma\}$, the inference problem is to find the best annotation $\mathbf{y}^*$ for a new image $\mathbf{x}$, i.e. $\mathbf{y}^* = \arg\max_{\mathbf{y}} f_\theta(\mathbf{x}, \mathbf{y})$. The inference requires solving the following optimization problem:

$$\max_{\mathbf{y} \in \mathcal{Y}} f_\theta(\mathbf{x}, \mathbf{y}) = \max_{s \in \mathcal{S}} \max_{\mathbf{y} \in \mathcal{Y}} \max_{\mathbf{z} \in \mathcal{Z}(\mathbf{y})} \theta^\top \Phi(\mathbf{x}, \mathbf{y}, \mathbf{z}, s) \tag{8}$$

Since we can enumerate all the possible values of the scene label $s$, the main difficulty of solving (8) is the inner maximization over $\mathbf{y}$ and $\mathbf{z}$ for a fixed $s$, i.e.:

$$\max_{\mathbf{y} \in \mathcal{Y}} \max_{\mathbf{z} \in \mathcal{Z}(\mathbf{y})} \theta^\top \Phi(\mathbf{x}, \mathbf{y}, \mathbf{z}, s) \tag{9}$$

In the following, we develop a method for solving (9) based on linear program (LP) relaxation. To formulate the problem as an LP, we first define the following:

$$a_{ij} = \sum_{c=1}^4 \sum_{w=1}^{N_c} \alpha_{w,j}^c \mathbb{1}(x_{ic} = w), \; \forall i, \forall j \qquad\qquad b_j = r_{j,1}^s - r_{j,0}^s, \; \forall j \tag{10}$$

Then it is easy to verify that the optimization problem in (9) can be equivalently written as (the constant in the objective not involving $\mathbf{y}$ or $\mathbf{z}$ is omitted):

$$\max_{\mathbf{y},\mathbf{z}} \sum_{i,j} a_{ij} z_{ij} + \sum_j b_j y_j \quad \text{s.t.} \quad \sum_j z_{ij} \leq 1, \ \max_i z_{ij} = y_j, \ z_{ij} \in \{0,1\}, \ \forall i \ \forall j \qquad (11)$$

The optimization problem (11) is not convex. But we can relax its constraints to make it an LP. First we reformulate (11) as an integer linear program (ILP):

$$\max_{\mathbf{y},\mathbf{z}} \sum_{i,j} a_{ij} z_{ij} + \sum_j b_j y_j \quad \text{s.t.} \sum_j z_{ij} \leq 1, \ z_{ij} \leq y_j \leq \sum_i z_{ij}, \ z_{ij} \in \{0,1\}, \ y_j \in \{0,1\}, \ \forall i \ \forall j \ (12)$$

It is easy to verify that (11) and (12) are equivalent. Of course, (12) still has the integral constraint $z_{ij} \in \{0,1\}$, which makes the optimization problem NP-hard. So we further relax the value of $z_{ij}$ to a real value in the range of $[0,1]$.

Putting everything together, the LP relaxation of (11) can be written as:

$$\max_{\mathbf{y},\mathbf{z}} \sum_{i,j} a_{ij} z_{ij} + \sum_j b_j y_j \quad \text{s.t.} \sum_j z_{ij} \leq 1, \ z_{ij} \leq y_j \leq \sum_i z_{ij}, \ 0 \leq z_{ij} \leq 1, \ 0 \leq y_j \leq 1, \ \forall i \ \forall j \ (13)$$

After solving (13) with any LP solver, we round $z_{ij}$ to the closest integer and obtain $y_j$ as $y_j = \max_i z_{ij}$.

## 4 Learning

We now describe how to learn the model parameters $\theta$ from a set of $N$ training examples $\langle \mathbf{x}^n, \mathbf{y}^n \rangle$ $(n = 1, 2, ..., N)$. Note that the training data only contain images and their annotations. We do not have the ground-truth scene label $s$ or the mapping $\mathbf{z}$ for any of the training images, so we have to treat them as latent variables during learning.

We adopt the latent SVM (LSVM) framework [7, 25] for learning. LSVMs extend the popular structural SVMs [18, 19] to handle latent variables during training. LSVMs and their variants have been successfully applied in several computer vision applications, e.g. object detection [7, 20], human action recognition [22, 16], human-object interaction [4], objects and attributes [23], human poses and actions [24], group activity recognition [9], etc.

The latent SVM learns the model parameters $\theta$ by solving the following optimization problem:

$$\min_{\theta} \frac{1}{2}||\theta||^2 + C \sum_{n=1}^{N} \xi_n \quad \text{s.t.} \quad f_\theta(\mathbf{x}^n, \mathbf{y}^n) - f_\theta(\mathbf{x}^n, \mathbf{y}) \geq \Delta(\mathbf{y}, \mathbf{y}^n) - \xi_n, \ \forall n, \ \forall y \qquad (14)$$

where $\Delta(\mathbf{y}, \mathbf{y}^n)$ is a loss function measuring the cost incurred by predicting $\mathbf{y}$ when the ground-truth annotation is $\mathbf{y}^n$. We use a simple Hamming loss which decomposes as $\Delta(\mathbf{y}, \mathbf{y}^n) = \sum_{j=1}^{V} \ell(y_j, y_j^n)$, where $\ell(y_j, y_j^n)$ is 1 if $y_j \neq y_j^n$ and 0 otherwise. Note that our loss function only involves the annotation $\mathbf{y}$, because this is the only ground-truth label we have access to.

The problem in (14) can be equivalently written as an unconstrained problem:

$$\min_{\theta} \frac{1}{2}||\theta||^2 + C \sum_{n=1}^{N} (\mathcal{L}^n - \mathcal{R}^n), \ \text{where } \mathcal{L}^n = \max_y \Big( \Delta(\mathbf{y}, \mathbf{y}^n) + f_\theta(\mathbf{x}^n, \mathbf{y}) \Big), \ \mathcal{R}^n = f_\theta(\mathbf{x}^n, \mathbf{y}^n) \ (15)$$

We use the non-convex bundle optimization in [5] to solve (15). In a nutshell, the algorithm iteratively builds an increasingly accurate piecewise quadratic approximation to the objective function. During each iteration, a new linear cutting plane is found via a subgradient of the objective function and added to the piecewise quadratic approximation. The key of applying this algorithm to solve (15) is computing the two subgradients $\partial_\theta \mathcal{L}^n$ and $\partial_\theta \mathcal{R}^n$ for a particular $\theta$, which we describe in detail below.

First we describe how to compute $\partial_\theta \mathcal{L}$. Let $(\mathbf{y}^*, \mathbf{z}^*, s^*)$ be the solution to the following optimization problem (called loss-augmented inference in the structural SVM literature):

$$\max_s \max_\mathbf{y} \max_{\mathbf{z} \in \mathcal{Z}(\mathbf{y})} \Delta(\mathbf{y}, \mathbf{y}^n) + f_\theta(\mathbf{x}^n, \mathbf{y}) \qquad (16)$$

Then it is easy to show that a subgradient $\partial_\theta \mathcal{L}^n$ can be calculated as $\partial_\theta \mathcal{L}^n = \Phi(\mathbf{x}^n, \mathbf{y}^*, \mathbf{z}^*, s^*)$. The loss-augmented inference problem in (16) is similar to the inference problem in (8), except for an additional term $\Delta(\mathbf{y}, \mathbf{y}^n)$. We can modify the LP relaxation method in Sec. 3 to solve (16) for a fixed $s$ (and enumerate $s$ to get the final solution). First of all, it is easy to verify that $\ell(y_j, y_j^n)$ can be re-formulated as:

$$\ell(y_j, y_j^n) \equiv \begin{cases} 1 - y_j & \text{if } y_j^n = 1 \\ y_j & \text{if } y_j^n = 0 \end{cases} \tag{17}$$

Using (17), it is easy to show that if we re-define $b_j$ as below, the ILP in (12) will solve the loss-augmented inference (16) for a fixed $s$:

$$b_j = \begin{cases} \gamma_{j,1}^s - \gamma_{j,0}^s - 1 & \text{if } y_j^n = 1 \\ \gamma_{j,1}^s - \gamma_{j,0}^s + 1 & \text{if } y_j^n = 0 \end{cases} \tag{18}$$

Similarly, we can relax the problem to an LP using the same method in Sec. 3.

Now we describe how to compute $\partial_\theta \mathcal{R}$. Let $(\mathbf{z}^\star, s^\star)$ be the solution to the following optimization problem: $\max_s \max_{\mathbf{z} \in \mathcal{Z}(\mathbf{y}^n)} f_\theta(\mathbf{x}^n, \mathbf{y}^n)$. Then it can be shown that a subgradient $\partial_\theta \mathcal{R}^n$ can be calculated as $\partial_\theta \mathcal{R}^n = \Phi(\mathbf{x}^n, \mathbf{y}^n, \mathbf{z}^\star, s^\star)$. For a fixed $s$, it is easy to show that the maximization over $\mathbf{z}$ can be solved by the following ILP:

$$\max_{\mathbf{z}} \sum_{i,j} a_{ij} z_{ij}, \ \text{s.t.} \ \sum_j z_{ij} = y_j^n, \ \forall i; \ z_{ij} \in \{0,1\}, \ \forall i \ \forall j \tag{19}$$

Similarly, we can solve (19) via LP relaxation by replacing the integral constraint $z_{ij} \in \{0,1\}$ with a linear constraint $0 \le z_{ij} \le 1$.

## 5 Experiments

We test our model on the UIUC sport dataset [11]. It contains images collected from eight sport classes: badminton, bocce, croquet, polo, rock climbing, rowing, sailing and snowboarding. Each image is annotated with a set of tags denoting the objects in it. We remove annotation terms occurring fewer than three times. We randomly choose half of the data as the test set. From the other half, we randomly select 50 images from each class to form the validation set. The remaining data are used as the training set.

We feed the training images and associated annotations (but not the ground-truth sport category labels) to our learning algorithm and set the number of latent scene labels to be eight (i.e. the number of sport classes). We initialize the parameters of our model as follows. First we cluster the training images into eight cluster using the following method. For each training image, we construct a feature vector from the visual information of the image itself and the textual information of its annotation. The visual information is simply the concatenation of visual word counts from all the regions in the image (normalized between 0 and 1), i.e. the dimensionality of the visual feature is $\sum_{c=1}^C N_c$. The textual information is the 0-1 vector of the annotation, i.e. the dimensionality is $V$. We then run k-means clustering based on the combined visual and textual features to cluster training images into eight clusters. We use the cluster membership of each training image as the initial guess of the scene label $s$ (which we call *pseudo-scene label*). We then initialize the parameters $\beta$ by examining the co-occurrence counts of visual words and pseudo-scene labels on the training data. Similarly, we initialize the parameters $\gamma$ by the co-occurrence counts of annotation terms and pseudo-scene labels. The parameters $\alpha$ are initialized by the co-occurrence counts of visual words and annotation terms with the mapping constraints ignored.

We compare our model with a baseline method which is a set of linear SVMs separately trained for predicting the 0/1 output of each annotation term based on the feature vector from the visual information. Following [21], we use the F-measure to measure the annotation performance. The comparison is shown in Table 1(a). Our model outperforms the baseline SVM method. We also list the published result of [22] in the table. However, it is important to remember that it is not directly comparable to other numbers in Table 1(a), since [22] uses different image features and different subsets of the dataset unspecified in the paper. We visualize some results on the test data in Fig. 5.

The scene labels $s$ produced by our model for the test images can be considered as a clustering of the scenes in those images. We can measure the quality of the scene clustering by comparing with

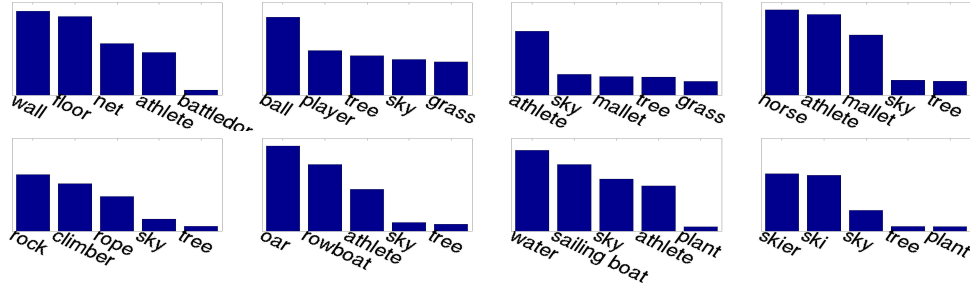

*Figure 3:* Visualization of $\gamma$ parameters. Each plot corresponds to a scene label $s$, we show the weights of top five components of $\gamma_{j,1}^s$ of all $j \in \{1..V\}$ (y-axis) and the corresponding annotation terms (x-axis).

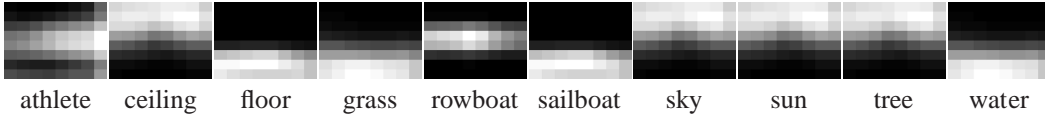

athlete    ceiling    floor    grass    rowboat  sailboat    sky    sun    tree    water

*Figure 4:* Visualization of the "position" components of the $\alpha$ parameters for some annotation terms. Bright areas correspond to high values.

the ground-truth scene labels (i.e. sport categories) of the test images. For comparison, we consider three baseline algorithms. The first baseline algorithm is to run k-means clustering on the test data based on the visual features. However the comparison to this baseline algorithm is not completely fair, since the baseline does not exploit any information from the annotations on the training data. So we define other two baseline algorithms that use this extra information.

For the second baseline algorithm (which we call *pseudo-label+SVM*), we run k-means clustering on both training and validation data. We use both visual features and textual features for the clustering. After running k-means clustering, we assign a *pseudo-label* to each image in the training or validation set by its cluster membership. Then we train a multi-class SVM based on the visual features of the training images and their pseudo-labels. The parameters of the SVM classifier are chosen by validating on the validation images (visual features only) with their pseudo-labels. For a test image, we use the trained SVM classifier to assign a pseudo-label based on the visual feature of this image. The predicted pseudo-labels of test images serve as a clustering of those images.

For the third baseline algorithm (which we call *pseudo-annotation+K-means*), we first train separate SVM classifiers to predict the annotation from the visual feature, using the ground-truth annotations of the validation set to choose the free parameters in SVM classifiers. For a set of test images, we use the trained SVM classifiers to predict their associated annotations (which we call *pseudo-annotations*). Then we run k-means to cluster those test images based on both visual features and textual features. The textual features are obtained from the pseudo-annotations.

We use the normalized mutual information (NMI) [15] to quantitatively measure the clustering results. Let $\Omega = \{\omega_1, \omega_2, ..., \omega_K\}$ be a set of clusters, and $\mathbb{D} = \{d_1, d_2, ..., d_K\}$ be the set of ground-truth categories. The NMI is defined as $\mathrm{NMI}(\Omega, \mathbb{D}) = \frac{I(\Omega;\mathbb{D})}{[H(\Omega)+H(\mathbb{D})]/2}$, where $I(\cdot)$ and $H(\cdot)$ are the mutual information and the entropy, respectively. The minimum of NMI is 0 if the cluster is random with respect to the ground-truth. Higher NMIs means better clustering results. The comparison is shown in Table 1(b). Our model outperforms other baseline methods on the scene clustering task.

We can visualize some of the parameters to get insights about the learned model. For a particular scene label $s$, the parameter $\gamma_{j,1}^s$ measures the compatibility of setting the $j$-th annotation term active for the scene label $s$. We sort the annotation terms according to $\gamma_{j,1}^s$. In Fig 3, we visualize the top five annotation terms for each of the eight possible values of $s$. Intuitively, these eight scene clusters obtained from our model seem to match well to the eight different sport categories of this dataset. We also visualize the "position" (i.e. $c = 4$) components of the $\alpha$ parameters (Fig. 4) for several annotation terms as follows. For a particular annotation term $j$, we find the most preferred "position" visual word $w^*$ for this annotation term by $w^* = \arg\max_w \alpha_{w,j}^4$. The cluster center of the visual word $w^*$ defines an $8 \times 8$ position mask of image locations (see [14]), which is visualized in Fig. 4. We can see that the learned $\alpha$ parameters make intuitive sense, e.g. "water" is preferred at the bottom of the image, while "sky" is preferred at the top of the image.

| method | F-measure |
|---|---|
| our approach | **0.4552** |
| SVM | 0.4112 |
| [21] | 0.3500 |

(a)

| method | NMI |
|---|---|
| our approach | **0.5295** |
| pseudo-label + SVM | 0.4134 |
| pseudo-annotation + K-means | 0.3267 |
| K-means | 0.2227 |

(b)

*Table 1:* Comparison of image annotation (a) and scene clustering (b). The number of clusters is set to be eight for all methods. See the text for more descriptions.

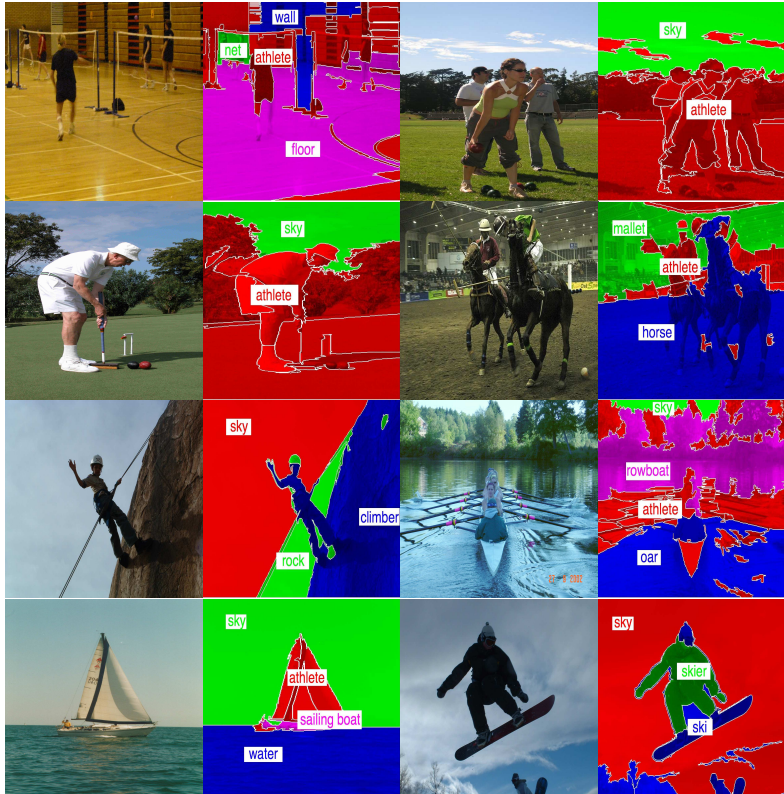

*Figure 5:* (Best viewed in color) Results of annotation and segmentation on the UIUC sport dataset. Different annotation terms are shown in different colors. Image regions mapped to an annotation term are overlayed with the color corresponding to that annotation term.

## 6  Conclusion

We have presented a discriminatively trained latent model for capturing the relationships among image regions, textual annotations, and overall scenes. Our ultimate goal is to achieve total scene understanding from cheaply available Internet data. Although most previous work in scene understanding focuses on generative probabilistic models (e.g. [1, 3, 11, 12, 21]), this paper offers an alternative path towards this goal via a discriminative framework. We believe discriminative methods offer a complementary advantage over generative ones. Certain relationships (e.g. the mapping between images regions and annotation terms) are hard to model, hence largely ignored in the generative approaches. But those relationships are easy to incorporate in a max-margin discriminative approach like ours.

In this work we have provided evidence that modeling these relationships can improve image annotation. Our work provides a general solution that can be broadly applied in other applications involving mapping relationships, e.g. Youtube videos with annotations, movie clips with captions, face detection with person names, etc. There are many open issues to address in future research: (1) extending our model to handle a richer set of annotation terms (nouns, verbs, adjectives, etc) by modifying the many-to-one correspondence assumption. (2) exploring the use of this model with noisier annotation data (e.g. raw Flickr or YouTube tags); (3) exploiting the linguistic structure of tags.

## Footnotes

\*Work done while the author was with Simon Fraser University.

# References

[1] K. Barnard, P. Duygulu, D. Forsyth, N. de Freitas, D. M. Blei, and M. I. Jordan. Matching words and pictures. *Journal of Machine Learning Research*, 3:1107–1135, 2003.

[2] K. Barnard and Q. Fan. Reducing correspondence ambiuity in loosely labeled training data. In *IEEE Computer Society Conference on Computer Vision and Pattern Recognition*, 2007.

[3] T. L. Berg, A. C. Berg, J. Edwards, and D. Forsyth. Who's in the picture. In *Advances in Neural Information Processing Systems*, volume 17, pages 137–144. MIT Press, 2004.

[4] C. Desai, D. Ramanan, and C. Fowlkes. Discriminative models for static human-object interactions. In *Workshop on Structured Models in Computer Vision*, 2010.

[5] T.-M.-T. Do and T. Artieres. Large margin training for hidden markov models with partially observed states. In *International Conference on Machine Learning*, 2009.

[6] M. Everingham, L. Van Gool, C. K. I. Williams, J. Winn, and A. Zisserman. The PASCAL visual object classes (VOC) challenge. *International Journal of Computer Vision*, 88(2):303–338, 2010.

[7] P. F. Felzenszwalb, R. B. Girshick, D. McAllester, and D. Ramanan. Object detection with discriminatively trained part based models. *IEEE Transactions on Pattern Analysis and Machine Intelligence*, 2009.

[8] P. F. Felzenszwalb and D. P. Huttenlocher. Efficient graph-based image segmentation. *International Journal of Computer Vision*, 2004.

[9] T. Lan, Y. Wang, W. Yang, and G. Mori. Beyond actions: Discriminative models for contextual group activities. In *Advances in Neural Information Processing Systems*. MIT Press, 2010.

[10] J. Li and J. Z. Wang. Automatic linguistic indexing of pictures by a statistical modeling approach. *IEEE Transactions on Pattern Analysis and Machine Intelligence*, 25(9):1075–1088, September 2003.

[11] L.-J. Li and L. Fei-Fei. What, where and who? classifying events by scene and object recognition. In *IEEE Computer Society Conference on Computer Vision and Pattern Recognition*, 2007.

[12] L.-J. Li, R. Socher, and L. Fei-Fei. Towards total scene understanding: Classification, annotation and segmentation in an automatic framework. In *IEEE Computer Society Conference on Computer Vision and Pattern Recognition*, 2009.

[13] N. Loeff and A. Farhadi. Scene discovery by matrix factorization. In *European Conference on Computer Vision*, 2008.

[14] T. Malisiewicz and A. A. Efros. Recognition by association via learning per-exemplar distances. In *IEEE Computer Society Conference on Computer Vision and Pattern Recognition*, 2008.

[15] C. D. Manning. *Introduction to Information Retrieval*. Cambridge University Press, 2008.

[16] J. C. Niebles, C.-W. Chen, and L. Fei-Fei. Modeling temporal structure of decomposable motion segments for activity classification. In *European Conference on Computer Vision*, 2010.

[17] R. Socher and L. Fei-Fei. Connecting modalities: Semi-supervised segmentation and annotation of images using unaligned text corpora. In *IEEE Computer Society Conference on Computer Vision and Pattern Recognition*, 2010.

[18] B. Taskar, C. Guestrin, and D. Koller. Max-margin markov networks. In *Advances in Neural Information Processing Systems*, volume 16. MIT Press, 2004.

[19] I. Tsochantaridis, T. Joachims, T. Hofmann, and Y. Altun. Large margin methods for structured and interdependent output variables. *Journal of Machine Learning Research*, 6:1453–1484, 2005.

[20] A. Vedaldi and A. Zisserman. Structured output regression for detection with partial truncation. In *Advances in Neural Information Processing Systems*. MIT Press, 2009.

[21] C. Wang, D. Blei, and L. Fei-Fei. Simultaneous image classification and annotation. In *IEEE Computer Society Conference on Computer Vision and Pattern Recognition*, 2009.

[22] Y. Wang and G. Mori. Max-margin hidden conditional random fields for human action recognition. In *IEEE Computer Society Conference on Computer Vision and Pattern Recognition*, 2009.

[23] Y. Wang and G. Mori. A discriminative latent model of object classes and attributes. In *European Conference on Computer Vision*, 2010.

[24] W. Yang, Y. Wang, and G. Mori. Recognizing human actions from still images with latent poses. In *IEEE Computer Society Conference on Computer Vision and Pattern Recognition*, 2010.

[25] C.-N. Yu and T. Joachims. Learning structural SVMs with latent variables. In *International Conference on Machine Learning*, 2009.

